# Contextual models for object detection using boosted random fields

**Antonio Torralba**
MIT, CSAIL
Cambridge, MA 02139
torralba@mit.edu

**Kevin P. Murphy**
UBC, CS
Vancouver, BC V6T 1Z4
murphyk@cs.ubc.edu

**William T. Freeman**
MIT, CSAIL
Cambridge, MA 02139
billf@mit.edu

## Abstract

We seek to both detect and segment objects in images. To exploit both local image data as well as contextual information, we introduce Boosted Random Fields (BRFs), which uses Boosting to learn the graph structure and local evidence of a conditional random field (CRF). The graph structure is learned by assembling graph fragments in an additive model. The connections between individual pixels are not very informative, but by using dense graphs, we can pool information from large regions of the image; dense models also support efficient inference. We show how contextual information from other objects can improve detection performance, both in terms of accuracy and speed, by using a computational cascade. We apply our system to detect stuff and things in office and street scenes.

## 1 Introduction

Our long-term goal is to build a vision system that can examine an image and describe what objects are in it, and where. In many images, such as Fig. 5(a), objects of interest, such as the keyboard or mouse, are so small that they are impossible to detect just by using local features. Seeing a blob next to a keyboard, humans can infer it is likely to be a mouse; we want to give a computer the same abilities.

There are several pieces of related work. Murphy et al [9] used global scene context to help object recognition, but did not model relationships between objects. Fink and Perona [4] exploited local dependencies in a boosting framework, but did not allow for multiple rounds of communication between correlated objects. He et al [6] do not model connections between objects directly, but rather they induce such correlations indirectly, via a bank of hidden variables, using a "restricted Boltzmann machine" architecture.

In this paper, we exploit contextual correlations between the object classes by introducing Boosted Random Fields (BRFs). Boosted random fields build on both boosting [5, 10] and conditional random fields (CRFs) [8, 7, 6]. Boosting is a simple way of sequentially constructing "strong" classifiers from "weak" components, and has been used for single-class object detection with great success [12]. Dieterich et al [3] combine boosting and 1D CRFs, but they only consider the problem of learning the local evidence potentials; we consider the much harder problem of learning the structure of a 2D CRF.

Standard applications of MRFs/ CRFs to images [7] assume a 4-nearest neighbor grid structure. While successful in low-level vision, this structure will fail in capturing important long distance dependencies between whole regions and across classes. We propose a method for learning densely connected random fields with long range connections. The

topology of these connections is chosen by a weak learner which has access to a library of graph fragments, derived from patches of labeled training images, which reflect typical spatial arrangments of objects (similar to the segmentation fragments in [2]). At each round of the learning algorithm, we add more connections from other locations in the image and from other classes (detectors). The connections are assumed to be spatially invariant, which means this update can be performed using convolution followed by a sigmoid nonlinearity. The resulting architecture is similar to a convolutional neural network, although we used a stagewise training procedure, which is much faster than back propagation.

In addition to recognizing things, such as cars and people, we are also interested in recognizing spatially extended "stuff" [1], such as roads and buildings. The traditional sliding window approach to object detection does not work well for detecting "stuff". Instead, we combine object detection and image segmentation (c.f., [2]) by labeling every pixel in the image. We do not rely on a bottom-up image segmentation algorithm, which can be fragile without top-down guidance.

## 2  Learning potentials and graph structure

A conditional random field (CRF) is a distribution of the form

$$P(S|x) = \frac{1}{Z} \prod_i \phi_i(S_i) \prod_{j \in N_i} \psi_{i,j}(S_i, S_j)$$

where $x$ is the input (e.g., image), $N_i$ are the neighbors of node $i$, and $S_i$ are labels. We have assumed pairwise potentials for notational simplicity. Our goal is to learn the local evidence potentials, $\phi_i$, the compatibility potentials $\psi$, and the set of neighbors $N_i$.

We propose the following simple approximation: use belief propagation (BP) to estimate the marginals, $P(S_i|x)$, and then use boosting to maximize the likelihood of each node's training data with respect to $\phi_i$ and $\psi$.

In more detail, the algorithm is as follows. At iteration $t$, the goal is to minimize the negative log-likelihood of the training data. As in [11], we consider the per-label loss (i.e., we use marginal probabilities), as opposed to requiring that the joint labeling be correct (as in Viterbi decoding). Hence the cost function to be minimized is

$$J^t = \prod_i J_i^t = - \prod_m \prod_i b_{i,m}^t(S_{i,m}) = - \prod_m \prod_i b_{i,m}^t(+1)^{S_{i,m}^*} b_{i,m}^t(-1)^{1-S_{i,m}^*} \quad (1)$$

where $S_{i,m} \in \{-1, +1\}$ is the true label for pixel $i$ in training case $m$, $S_{i,m}^* = (S_{i,m} + 1)/2 \in \{0, 1\}$ is just a relabeling, and $b_{i,m}^t = [P(S_i = -1|x_m, t), P(S_i = 1|x_m, t)]$ is the belief state at node $i$ given input image $x_m$ after $t$ iterations of the algorithm.

The belief at node $i$ is given by the following (dropping the dependence on case $m$) $b_i^t(\pm 1) \propto \phi_i^t(\pm 1) M_i^t(\pm 1)$ where $M_i^t$ is the product of all the messages coming into $i$ from all its neighbors at time $t$ and where the message that $k$ sends to $i$ is given by

$$M_i^{t+1}(\pm 1) = \prod_{k \in N_i} \mu_{k \to i}^{t+1}(\pm 1) \qquad \mu_{k \to i}^{t+1}(\pm 1) = \sum_{s_k \in \{-1, +1\}} \psi_{k,i}(s_k, \pm 1) \frac{b_k^t(s_k)}{\mu_{i \to k}^t(s_k)} \quad (2)$$

where $\psi_{k,i}$ is the compatibility between nodes $k$ and $i$. If we assume that the local potentials have the form $\phi_i^t(s_i) = [e^{F_i^t/2}; e^{-F_i^t/2}]$, where $F_i^t$ is some function of the input data, then:

$$b_i^t(+1) = \sigma(F_i^t + G_i^t), \qquad G_i^t = \log M_i^t(+1) - \log M_i^t(-1) \quad (3)$$

where $\sigma(u) = 1/(1 + e^{-u})$ is the sigmoid function. Hence each term in Eq. 1 simplifies to a cost function similar to that used in boosting:

$$\log J_i^t = \sum_m \log \left( 1 + e^{-S_{i,m}(F_{i,m}^t + G_{i,m}^t)} \right). \quad (4)$$

1. **Input**: a set of labeled pairs $\{x_{i,m}; S_{i,m}\}$, bound $T$
   **Output**: Local evidence functions $f_i^t(x)$ and message update functions $g_i^t(b_{N_i})$.

2. Initialize: $b_{i,m}^{t=0} = 0$; $F_{i,m}^{t=0} = 0$; $G_{i,m}^{t=0} = 0$

3. For t=1..T.

   (a) Fit local potential $f_i(x_{i,m})$ by weighted LS to
   $$Y_{i,m}^t = S_{i,m}(1 + e^{-S_{i,m}(F_i^t + G_{i,m}^t)})$$

   (b) Fit compatibilities $g_i^t(b_{N_{i,m}}^{t-1})$ to $Y_{i,m}^t$ by weighted LS.

   (c) Compute local potential $F_{i,m}^t = F_{i,m}^{t-1} + f_i^t(x_{i,m})$

   (d) Compute compatibilities $G_{i,m}^t = \sum_{n=1}^t g_i^n(b_{N_{i,m}}^{t-1})$

   (e) Update the beliefs $b_{i,m}^t = \sigma(F_{i,m}^t + G_{i,m}^t)$

   (f) Update weights $w_{i,m}^{t+1} = b_{i,m}^t(-1)\, b_{i,m}^t(+1)$

Figure 1: BRF training algorithm.

We assume that the graph is very densely connected so that the information that one single node sends to another is so small that we can make the approximation $\mu_{k\rightarrow i}^{t+1}(+1)/\mu_{k\rightarrow i}^{t+1}(-1) \simeq 1$. (This is a reasonable approximation in the case of images, where each node represents a single pixel; only when the influence of many pixels is taken into account will the messages become informative.) Hence

$$G_i^{t+1} = \log \frac{M_i^{t+1}(+1)}{M_i^{t+1}(-1)} = \sum_k \log \frac{\sum_{s_k \in [-1,+1]} \psi_{k,i}(s_k, +1) \frac{b_{k,m}^t(s_k)}{\mu_{i\rightarrow k}^t(s_k)}}{\sum_{s_k \in [-1,+1]} \psi_{k,i}(s_k, -1) \frac{b_{k,m}^t(s_k)}{\mu_{i\rightarrow k}^t(s_k)}} \tag{5}$$

$$\simeq \sum_k \log \frac{\sum_{s_k \in [-1,+1]} \psi_{k,i}(s_k, +1)\, b_{k,m}^t(s_k)}{\sum_{s_k \in [-1,+1]} \psi_{k,i}(s_k, -1)\, b_{k,m}^t(s_k)} \tag{6}$$

With this simplification, $G_i^{t+1}$ is now a non-linear function of the beliefs $G_i^{t+1}(\vec{b}_m^t)$ at iteration $t$. Therefore, We can write the beliefs at iteration $t$ as a function of the local evidences and the beliefs at time $t-1$: $b_i^t(+1) = \sigma(F_i^t(x_{i,m}) + G_i^t(\vec{b}_m^{t-1}))$. The key idea behind BRFs is to use boosting to learn the $G$ functions, which approximately implement message passing in densely connected graphs. We explain this in more detail below.

## 2.1 Learning local evidence potentials

Defining $F_i^t(x_{i,m}) = F_i^{t-1}(x_{i,m}) + f_i^t(x_{i,m})$ as an additive model, where $x_{i,m}$ are the features of training sample $m$ at node $i$, we can learn this function in a stagewise fashion by optimizing the second order Taylor expansion of Eq. 4 wrt $f_i^t$, as in logitBoost [5]:

$$\arg \min_{f_i^t} \log J_i^t \simeq \arg \min_{f_i^t} \sum_m w_{i,m}^t (Y_{i,m}^t - f_i^t(x_{i,m}))^2 \tag{7}$$

where $Y_{i,m}^t = S_{i,m}(1 + e^{-S_{i,m}(F_i^t + G_{i,m}^t)})$. In the case that the weak learner is a "regression stump", $f_i(x) = ah(x) + b$, we can find the optimal $a, b$ by solving a weighted least squares problem, with weights $w_{i,m}^t = b_i^t(-1)\, b_i^t(+1)$; we can find the best basis function $h(x)$ by searching over all elements of a dictionary.

## 2.2 Learning compatibility potentials and graph structure

In this section, we discuss how to learn the compatibility functions $\psi_{ij}$, and hence the structure of the graph. Instead of learning the compatibility functions $\psi_{ij}$, we propose to

1. **Input**: a set of inputs $\{x_{i,m}\}$ and functions $f_i^t$, $g_i^t$
   **Output**: Set of beliefs $b_{i,m}$ and MAP estimates $S_{i,m}$.

2. Initialize: $b_{i,m}^{t=0} = 0$; $F_{i,m}^{t=0} = 0$; $G_{i,m}^{t=0} = 0$

3. From $t = 1$ to $T$, repeat

   (a) Update local evidences $F_{i,m}^t = F_{i,m}^{t-1} + f_i^t(x_{i,m})$

   (b) Update compatibilities $G_{i,m}^t = \sum_{n=1}^t g_i^n(b_{N_i,m}^{t-1})$

   (c) Compute current beliefs $b_{i,m}^t = \sigma(F_{i,m}^t + G_{i,m}^t)$

4. Output classification is $S_{i,m} = \delta\left(b_{i,m}^t > 0.5\right)$

Figure 2: BRF run-time inference algorithm.

learn directly the function $G_i^{t+1}$. We propose to use an additive model for $G_i^{t+1}$ as we did for learning $F$: $G_{i,m}^{t+1} = \sum_{n=1}^t g_i^n(\vec{b}_m^t)$, where $\vec{b}_m^t$ is a vector with the beliefs of all nodes in the graph at iteration $t$ for the training sample $m$. The weak learners $g_i^n(\vec{b}_m^t)$ can be regression stumps with the form $g_i^n(\vec{b}_m^t) = a\delta(\vec{w} \cdot \vec{b}_m^t > \theta) + b$, where $a, b, \theta$ are the parameters of the regression stump, and $\vec{w}_i$ is a set of weights selected from a dictionary. In the case of a graph with weak and almost symmetrical connections (which holds if $\psi(s_1, s_2) \approx 1$, for all $(s_1, s_2)$, which implies the messages are not very informative) we can further simplify the function $G_i^{t+1}$ by approximating it as a linear function of the beliefs:

$$G_{i,m}^{t+1} = \sum_{k \in N_i} \alpha_{k,i} \, b_{k,m}^t(+1) + \beta_{k,i} \tag{8}$$

This step reduces the computational cost. The weak learners $g_i^n(\vec{b}_m^t)$ will also be linear functions. Hence the belief update simplifies to $b_{i,m}^{t+1}(+1) = \sigma(\vec{\alpha}_i \cdot \vec{b}_m^t + \beta_i + F_{i,m}^t)$, which is similar to the mean-field update equations. The neighborhood $N_i$ over which we sum incoming messages is determined by the graph structure, which is encoded in the non-zero values of $\alpha_i$. Each weak learner $g_i^n$ will compute a weighted combination of the beliefs of the some subset of the nodes; this subset may change from iteration to iteration, and can be quite large. At iteration $t$, we choose the weak learner $g_i^t$ so as to minimize

$$\log J_i^t(b^{t-1}) = -\sum_m \log\left(1 + e^{-S_{i,m}(F_{i,m}^t + g_i^t(b_m^{t-1}) + \sum_{n=1}^{t-1} g_i^n(b_m^{t-1}))}\right)$$

which reduces to a weighted least squares problem similar to Eq. 7. See Fig. 1 for the pseudo-code for the complete learning algorithm, and Fig. 2 for the pseudo-code for run-time inference.

## 3 BRFs for multiclass object detection and segmentation

With the BRF training algorithm in hand, we describe our approach for multiclass object detection and region-labeling using densely connected BRFs.

### 3.1 Weak learners for detecting stuff and things

The square sliding window approach does not provide a natural way of working with irregular objects. Using region labeling as an image representation allows dealing with irregular and extended objects (buildings, bookshelf, road, ...). Extended stuff [1] may be a very important source of contextual information for other objects.

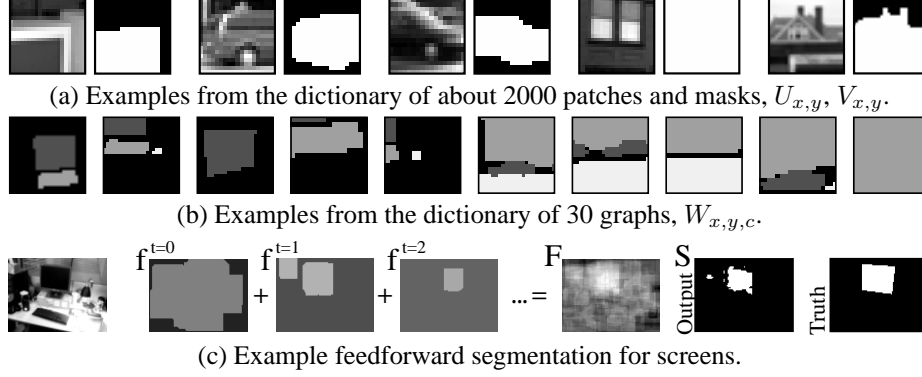

(a) Examples from the dictionary of about 2000 patches and masks, $U_{x,y}$, $V_{x,y}$.

(b) Examples from the dictionary of 30 graphs, $W_{x,y,c}$.

(c) Example feedforward segmentation for screens.

Figure 3: Examples of patches from the dictionary and an example of the segmentation obtained using boosting trained with patches from (a).

The weak learners we use for the local evidence potentials are based on the segmentation fragments proposed in [2]. Specifically, we create a dictionary of about 2000 image patches $U$, chosen at random (but overlapping each object), plus a corresponding set of binary (in-class/ out-of-class) image masks, $V$: see Fig. 3(a). At each round $t$, for each class $c$, and for each dictionary entry, we construct the following weak learner, whose output is a binary matrix of the same size as the image $I$:

$$v(I) = ((I \otimes U) > \theta) * V > 0 \tag{9}$$

where $\otimes$ represents normalized cross-correlation and $*$ represents convolution. The intuition behind this is that $I \otimes U$ will produce peaks at image locations that contain this patch/template, and then convolving with $V$ will superimpose the segmentation mask on top of the peaks. As a function of the threshold $\theta$, the feature will behave more as a template detector ($\theta \simeq 1$) or as a texture descriptor ($\theta << 1$).

To be able to detect objects at multiple scales, we first downsample the image to scale $\sigma$, compute $v(I \downarrow \sigma)$, and then upsample the result. The final weak learner does this for multiple scales, ORs all the results together, and then takes a linear transformation.

$$f(I) = \alpha \left( \vee_\sigma [v(I \downarrow \sigma) \uparrow \sigma] \right) + \beta \tag{10}$$

Fig. 3(c) shows an example of segmentation obtained by using boosting without context. The weak learners we use for the compatibility functions have a similar form:

$$g_c(b) = \alpha \left( \sum_{c'=1}^{C} b_{c'} * W_{c'} \right) + \beta \tag{11}$$

where $b_{c'}$ is the image formed by the beliefs at all pixels for class $c'$. This convolution corresponds to eq. 8 in which the node $i$ is one pixel $x, y$ of class $c$. The binary kernels (graph fragments) $W$ define, for each node $x, y$ of object class $c$, all the nodes from which it will receive messages. These kernels are chosen by sampling patches of various sizes from the labeling of images from the training set. This allows generating complicated patterns of connectivity that reflect the statistics of object co-occurrences in the training set. The overall incoming message is given by adding the kernels obtained at each boosting round. (This is the key difference from mutual boosting [4], where the incoming message is just the output of a single weak learner; thus, in mutual boosting, previously learned inter-class connections are only used once.) Although it would seem to take $O(t)$ time to compute $G^t$, we can precompute a single equivalent kernel $W'$, so at runtime the overall complexity is still linear in the number of boosting rounds, $O(T)$.

$$G_{x,y,c}^t = \sum_{c'=1}^{C} b_{c'} * \left( \sum_{n=1}^{t} \alpha^n W_{c'}^n \right) + \sum_n \beta^n \stackrel{\text{def}}{=} \sum_{c'=1}^{C} b_{c'} * W_{c'}' + \beta'$$

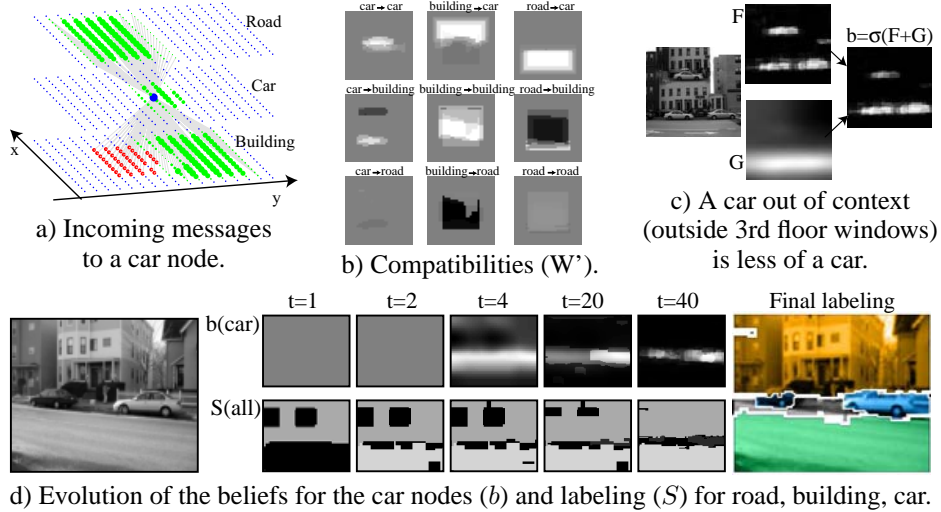

a) Incoming messages to a car node.

b) Compatibilities (W').

c) A car out of context (outside 3rd floor windows) is less of a car.

d) Evolution of the beliefs for the car nodes ($b$) and labeling ($S$) for road, building, car.

Figure 4: Street scene. The BRF is trained to detect cars, buildings and the road.

In Fig. 4(a-b), we show the structures of the graph and the weights $W'$ defined by $G_T$ for a BRF trained to detect cars, buildings and roads in street scenes.

## 3.2 Learning and inference

For training we used a labeled dataset of office and street scenes with about 100 images in each set. During the training, in the first 5 rounds we only update the local potentials, to allow local evidence to accrue. After the 5th iteration we start updating also the compatibility functions. At each round, we update only the local potential and compatibility function associated with a single object class that reduces the most the multiclass cost. This allows objects that need many features to have more complicated local potentials.

The algorithm learns to first detect easy (and large) objects, since these reduce the error of all classes the fastest. The easy-to-detect objects can then pass information to the harder ones. For instance, in office scenes, the system first detects screens, then keyboards, and finally computer mice. Fig. 5 illustrates this behavior on the test set. A similar behavior is obtained for the car detector (Fig. 4(d)). The detection of building and road provides strong constraints for the locations of the car.

## 3.3 Cascade of classifiers with BRFs

The BRF can be turned into a cascade [12] by thresholding the beliefs. Computations can then be reduced by doing the convolutions (required for computing $f$ and $g$) only in pixels that are still candidates for the presence of the target. At each round we update a binary rejection mask for each object class, $R_{x,y,c}^t$, by thresholding the beliefs at round $t$: $R_{x,y,c}^t = R_{x,y,c}^{t-1} \delta(b_{x,y,c}^t > \theta_c^t)$. A pixel in the rejection mask is set to zero when we can decide that the object is not present (when $b_{x,y,c}^t$ is below the threshold $\theta_c^t \simeq 0$), and it is set to 1 when more processing is required. The threshold $\theta_c^t$ is chosen so that the percentage of missed detections is below a predefined level (we use $1\%$). Similarity we can define a detection mask that will indicate pixels in which we decide the object is present. The mask is then used for computing the features $v(I)$ and messages $G$ by applying the convolutions only on the pixels not yet classified. We can denote those operators as $\otimes_R$ and $*_R$. This

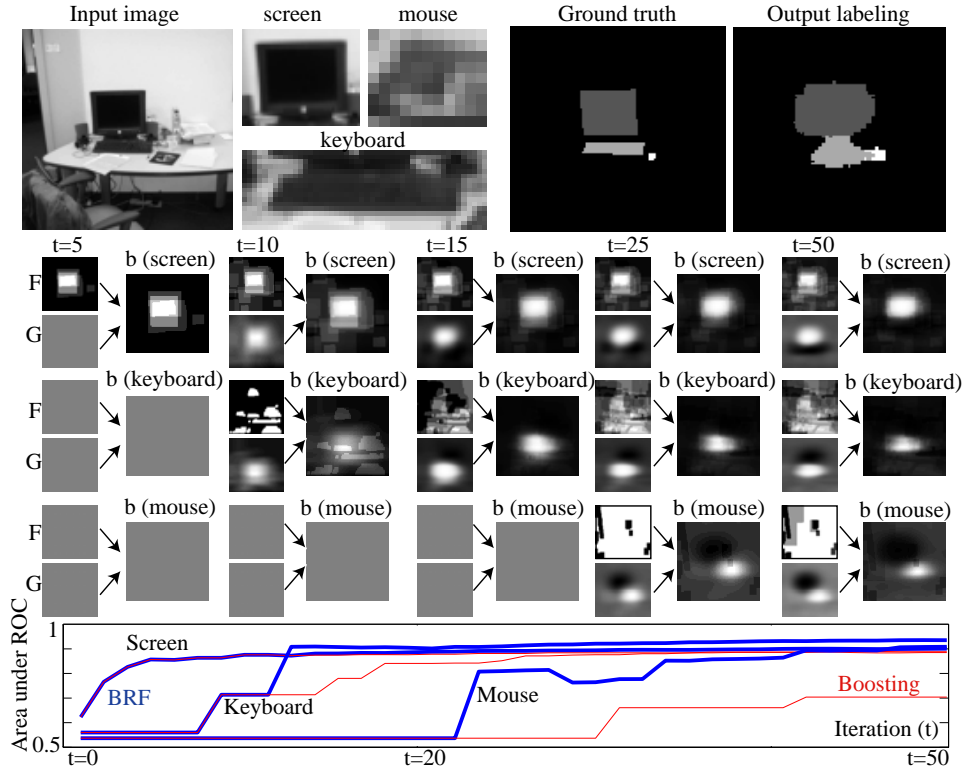

Figure 5: **Top**. In this desk scene, it is easy to identify objects like the screen, keyboard and mouse, even though the local information is sometimes insufficient. **Middle**: the evolution of the beliefs ($b$ and $F$ and $G$) during detection for a test image. **Bottom**. The graph bellow shows the average evolution of the area under the ROC for the three objects on 120 test images.

results in a more efficient classifier with only a slight decrease of performance. In Fig. 6 we compare the reduction of the search space when implementing a cascade using independent boosting (which reduces to Viola and Jones [12]), and when using BRF's. We see that for objects for which context is the main source of information, like the mouse, the reduction in search space is much more dramatic using BRFs than using boosting alone.

## 4    Conclusion

The proposed BRF algorithm combines boosting and CRF's, providing an algorithm that is easy for both training and inference. We have demonstrated object detection in cluttered scenes by exploiting contextual relationships between objects. The BRF algorithm is computationally efficient and provides a natural extension of the cascade of classifiers by integrating evidence from other objects in order to quickly reject certain image regions. The BRF's densely connected graphs, which efficiently collect information over large image regions, provide an alternative framework to nearest-neighbor grids for vision problems.

**Acknowledgments**

This work was sponsored in part by the Nippon Telegraph and Telephone Corporation as part of the NTT/MIT Collaboration Agreement, by BAE systems, and by DARPA contract DABT63-99-1-0012.

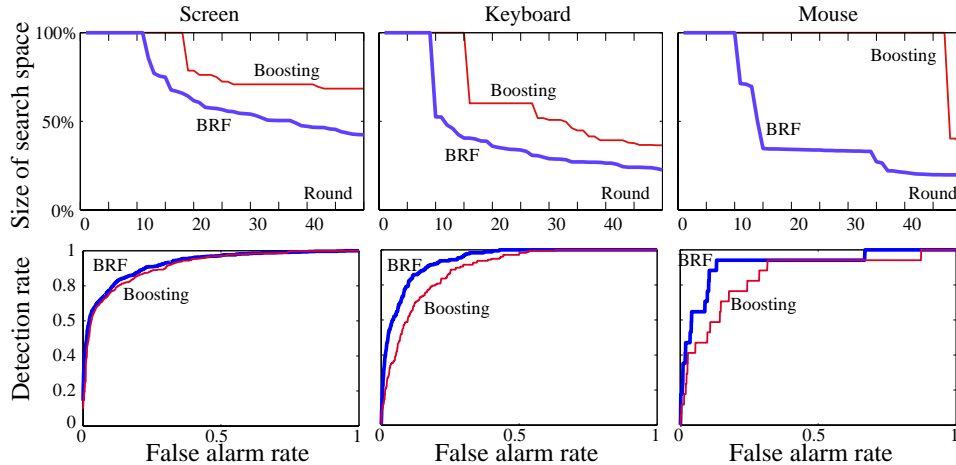

Figure 6: Contextual information reduces the search space in the framework of a cascade and improves performances. The search space is defined as the percentage of pixels that require further processing before a decision can be reached at each round. BRF's provide better performance and requires fewer computations. The graphs (search space and ROCs) correspond to the average results on a test set of 120 images.

# References

[1] E. H. Adelson. On seeing stuff: the perception of materials by humans and machines. In *Proc. SPIE*, volume 4299, pages 1–12, 2001.

[2] E. Borenstein and S. Ullman. Class-specific, top-down segmentation. In *Proc. European Conf. on Computer Vision*, 2002.

[3] T. Dietterich, A. Ashenfelter, and Y. Bulatov. Training conditional random fields via gradient tree boosting. In *Intl. Conf. on Machine Learning*, 2004.

[4] M. Fink and P. Perona. Mutual boosting for contextual influence. In *Advances in Neural Info. Proc. Systems*, 2003.

[5] J. Friedman, T. Hastie, and R. Tibshirani. Additive logistic regression: a statistical view of boosting. *Annals of statistics*, 28(2):337–374, 2000.

[6] Xuming He, Richard Zemel, and Miguel Carreira-Perpinan. Multiscale conditional random fields for image labelling. In *Proc. IEEE Conf. Computer Vision and Pattern Recognition*, 2004.

[7] S. Kumar and M. Hebert. Discriminative random fields: A discriminative framework for contextual interaction in classification. In *IEEE Conf. on Computer Vision and Pattern Recognition*, 2003.

[8] J. Lafferty, A. McCallum, and F. Pereira. Conditional random fields: Probabilistic models for segmenting and labeling sequence data. In *Intl. Conf. on Machine Learning*, 2001.

[9] K. Murphy, A. Torralba, and W. Freeman. Using the forest to see the trees: a graphical model relating features, objects and scenes. In *Advances in Neural Info. Proc. Systems*, 2003.

[10] R. Schapire. The boosting approach to machine learning: An overview. In *MSRI Workshop on Nonlinear Estimation and Classification*, 2001.

[11] B. Taskar, C. Guestrin, and D. Koller. Max-margin markov networks. In *Advances in Neural Info. Proc. Systems*, 2003.

[12] P. Viola and M. Jones. Robust real-time object detection. *Intl. J. Computer Vision*, 57(2):137–154, 2004.
